# Software for ANN training on a Ring Array Processor

**Phil Kohn, Jeff Bilmes, Nelson Morgan, James Beck**
International Computer Science Institute,
1947 Center St., Berkeley CA 94704, USA

## Abstract

Experimental research on Artificial Neural Network (ANN) algorithms requires either writing variations on the same program or making one monolithic program with many parameters and options. By using an object-oriented library, the size of these experimental programs is reduced while making them easier to read, write and modify. An efficient and flexible realization of this idea is Connectionist Layered Object-oriented Network Simulator (CLONES). CLONES runs on UNIX[1] workstations and on the 100-1000 MFLOP Ring Array Processor (RAP) that we built with ANN algorithms in mind. In this report we describe CLONES and show how it is implemented on the RAP.

## 1 Overview

As we continue to experiment with Artificial Neural Networks (ANNs) to generate phoneme probabilities for speech recognition (Bourlard & Morgan, 1991), two things have become increasingly clear:

1. Because of the diversity and continuing evolution of ANN algorithms, the programming environment must be both powerful and flexible.

2. These algorithms are very computationally intensive when applied to large databases of training patterns.

Ideally we would like to implement and test ideas at about the same rate that we come up with them. We have approached this goal both by developing application specific parallel

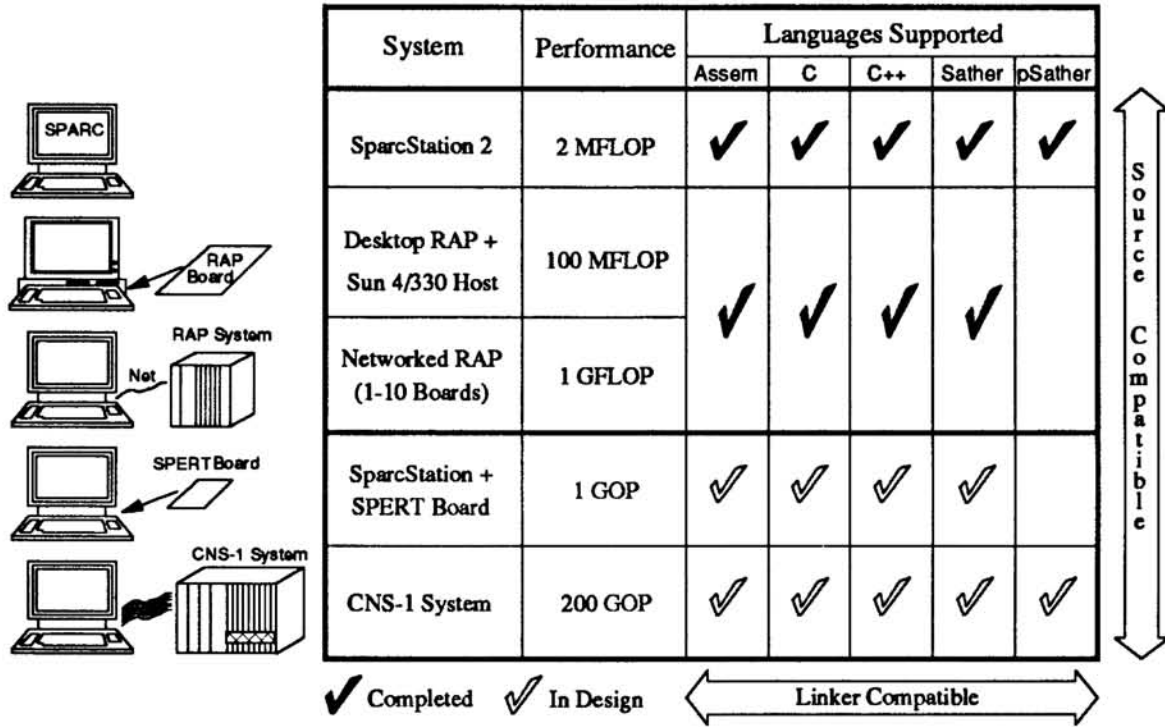

| System | Performance | Languages Supported | | | | |
|---|---|---|---|---|---|---|
| | | Assem | C | C++ | Sather | pSather |
| SparcStation 2 | 2 MFLOP | ✓ | ✓ | ✓ | ✓ | ✓ |
| Desktop RAP + Sun 4/330 Host | 100 MFLOP | ✓ | ✓ | ✓ | ✓ | |
| Networked RAP (1-10 Boards) | 1 GFLOP | | | | | |
| SparcStation + SPERT Board | 1 GOP | ✓ | ✓ | ✓ | ✓ | |
| CNS-1 System | 200 GOP | ✓ | ✓ | ✓ | ✓ | ✓ |

✓ Completed    ✓ In Design    Linker Compatible

Source Compatible

Figure 1: Hardware and software configurations

hardware, the Ring Array Processor (RAP) (Morgan et al., 1990; Beck, 1990; Morgan et al., 1992), and by building an object-oriented software environment, the Connectionist Layered Object-oriented Network Simulator (CLONES) (Kohn, 1991). By using an object-oriented library, the size of experimental ANN programs can be greatly reduced while making them easier to read, write and modify. CLONES is written in C++ and utilizes libraries previously written in C and assembler.

Our ANN research currently encompasses two hardware platforms and several languages, shown in Figure 1. Two new hardware platforms, the SPERT board (Asanović et al., 1991) and the CNS-1 system are in design (unfilled check marks), and will support source code compatibility with the existing machines. The SPERT design is a custom VLSI parallel processor installed on an SBUS card plugged into a SPARC workstation. Using variable precision fixed point arithmetic, a single SPERT board will have performance comparable to a 10 board RAP system with 40 processors. The CNS-1 system is based on multiple VLSI parallel processors interconnected by high speed communication rings.

Because the investment in software is generally large, we insist on source level compatibility across hardware platforms at the level of the system libraries. These libraries include matrix and vector classes that free the user from concern about the hardware configuration. It is also considered important to allow routines in different languages to be linked together. This includes support for Sather, an object-oriented language that has been developed at ICSI for workstations. The parallel version of Sather, called pSather, will be supported on

the CNS-1.

CLONES is seen as the ANN researcher's interface to this multiplatform, multilanguage environment. Although CLONES is an application written specifically for ANN algorithms, it's object-orientation gives it the ability to easily include previously developed libraries. CLONES currently runs on UNIX workstations and the RAP; this paper focuses on the RAP implementation.

## 2   RAP hardware

The RAP consists of cards that are added to a UNIX host machine (currently a VME based Sun SPARC). A RAP card has four 32 MFlop Digital Signal Processor (DSP) chips (TI TMS320C30), each with its own local 256KB or 1MB of fast static RAM and 16MB of DRAM.

Instead of sharing memory, the processors communicate on a high speed ring that shifts data in a single machine cycle. For each board, the peak transfer rate between 4 nodes is 64 million words/sec (256 Mbytes/second). This is a good balance to the 64 million multiply-accumulates per second (128 MFLOPS) peak performance of the computational elements.

Up to 16 of these boards can be interconnected and used as one Single Program operating on Multiple Data stream (SPMD) machine. In this style of parallel computation, all the processors run the same program and are doing the same operations to different pieces of the same matrix or vector [2]. The RAP can run other styles of parallel computation, including pipelines where each processor is doing a different operation on different data streams. However, for fully connected back-propagation networks, SPMD parallelism works well and is also much easier to program since there is only one flow of control to worry about.

A reasonable design for networks in which all processors need all unit outputs is a single broadcast bus. However, this design is not appropriate for other related algorithms such as the backward phase of the back-propagation learning algorithm. By using a ring, back-propagation can be efficiently parallelized without the need to have the complete weight matrix on all processors. The number of ring operations required for each complete matrix update cycle is of the same order as the number of units, **not** the square of the number of units. It should also be noted that we are using a stochastic or on-line learning algorithm. The training examples are **not** dividing among the processors then the weights batch updated after a complete pass. All weights are updated for each training example. This procedure greatly decreases the training time for large redundant training sets since more steps are being taken in the weight-space per training example.

We have empirically derived formulae that predict the performance improvement on back-propagation training as a function of the number of boards. Theoretical peak performance is 128 MFlops/board, with sustained performance of 30-90% for back-propagation problems of interest to us. Systems with up to 40 nodes have been tested, for which throughputs

of up to 574 Million Connections Per Second (MCPS) have been measured, as well as learning rates of up to 106 Million Connection Updates Per Second (MCUPS) for training. Practical considerations such as workstation address space and clock skew restrict current implementations to 64 nodes, but in principle the architecture scales to about 16,000 nodes for back-propagation.

We now have considerable experience with the RAP as a day-to-day computational tool for our research. With the aid of the RAP hardware and software, we have done network training studies that would have over a century on a UNIX workstation such as the SPARCstation-2. We have also used the RAP to simulate variable precision arithmetic to guide us in the design of higher performance hardware such as SPERT.

The RAP hardware remains very flexible because of the extensive use of programmable logic arrays. These parts are automatically downloaded when the host machine boots up. By changing the download files, the functionality of the communications ring and the host interface can be modified or extended without any physical changes to the board.

## 3   RAP software

The RAP DSP software is built in three levels (Kohn & Bilmes, 1990; Bilmes & Kohn, 1990). At the lowest level are hand coded assembler routines for matrix, vector and ring operations. Many standard matrix and vector operations are currently supported as well as some operations specialized for efficient back-propagation. These matrix and vector routines do not use the communications ring or split up data among processing nodes. There is also a UNIX compatible library including most standard C functions for file, math and string operations. All UNIX kernel calls (such as file input or output) cause requests to be made to the host SPARC over the VMEbus. A RAP dæmon process running under UNIX has all of the RAP memory mapped into its virtual address space. It responds to the RAP system call interrupts (from the RAP device driver) and can access RAP memory with a direct memory copy function or assignment statement.

An intermediate level consists of matrix and vector object classes coded in C++. A programmer writing at this level or above can program the RAP as if it were a conventional serial machine. These object classes divide the data and processing among the available processing nodes, using the communication ring to redistribute data as needed. For example, to multiply a matrix by a vector, each processor would have its own subset of the matrix rows that must be multiplied. This is equivalent to partitioning the output vector elements among the processors. If the complete output vector is needed by all processors, a ring broadcast routine is called to redistribute the part of the output vector from each processor to all the other processors.

The top level of RAP software is the CLONES environment. CLONES is an object-oriented library for constructing, training and utilizing connectionist networks. It is designed to run efficiently on data parallel computers as well as uniprocessor workstations. While efficiency and portability to parallel computers are the primary goals, there are several secondary design goals:

1. minimize the learning curve for using CLONES;

2. minimize the additional code required for new experiments;

3. maximize the variety of artificial neural network algorithms supported;

4. allow heterogeneous algorithms and training procedures to be interconnected and trained together;

5. allow the trained network to be easily embedded into other programs.

The size of experimental ANN programs is greatly reduced by using an object-oriented library; at the same time these programs are easier to read, write and evolve.

Researchers often generate either a proliferation of versions of the same basic program, or one giant program with a large number of options and many potential interactions and side-effects. Some simulator programs include (or worse, evolve) their own language for describing networks. We feel that a modern object-oriented language (such as C++) has all the functionality needed to build and train ANNs. By using an object-oriented design, we attempt to make the most frequently changed parts of the program very small and well localized. The parts that rarely change are in a centralized library. One of the many advantages of an object-oriented library for experimental work is that any part can be specialized by making a new class of object that inherits the desired operations from a library class.

## 4   CLONES overview

To make CLONES easier to learn, we restrict ourselves to a subset of the many features of C++. Excluded features include multiple inheritance, operator overloading (however, function overloading is used) and references. Since the multiple inheritance feature of C++ is not used, CLONES classes can be viewed as a collection of simple inheritance trees. This means that all classes of objects in CLONES either have no parent class (top of a class tree) or inherit the functions and variables of a single parent class.

CLONES consists of a library of C++ classes that represent networks (**Net**), their components (**Net_part**) and training procedures. There are also utility classes used during training such as: databases of training data (**Database**), tables of parameters and arguments (**Param**), and performance statistics (**Stats**). **Database** and **Param** do not inherit from any other class. Their class trees are independent of the rest of CLONES and each other. The **Stats** class inherits from **Net_behavior**.

The top level of the CLONES class tree is a class called **Net_behavior**. It defines function interfaces for many general functions including file save or restore and debugging. It also contains behavior functions that are called during different phases of running or training a network. For example, there are functions that are called before or after a complete training run (**pre_training**, **post_training**), before or after a pass over the database (**pre_epoch**, **post_epoch**) and before or after a forward or backward run of the network (**pre_forw_pass**, **post_forw_pass**, **pre_back_pass**, **post_back_pass**). The **Net**, **Net_part** and **Stats** classes inherit from this class.

All network components used to construct ANNs are derived from the two classes **Layer** and **Connect**. Both of these inherit from class **Net_part**. A CLONES network can be viewed as a graph where the nodes are **Layer** objects and the arcs are **Connect** objects. Each **Connect** connects a single input **Layer** with a single output **Layer**. A **Layer** holds the data for a set of units (such as an activation vector), while a **Connect** transforms the data as it passes between **Layers**. Data flows along **Connects** between the pair of **Layers** by calling **forw_propagate** (input to output) or **back_propagate** (output to input) behavior

functions in the **Connect** object.

CLONES does not have objects that represent single units (or artificial neurons). Instead, **Layer** objects are used to represent a set of units. Because arrays of units are passed down to the lowest level routines, most of the computation time is focused into a few small assembly coded loops that easily fit into the processor instruction cache. Time spent in all of the levels of control code that call these loops becomes less significant as the size of the **Layer** is increased.

The **Layer** class does not place any restrictions on the representation of its internal information. For example, the representation for activations may be a floating point number for each unit (**Analog_layer**), or it may be a set of unit indices, indicating which units are active (**Binary_layer**). **Analog_layer** and **Binary_layer** are built into the CLONES library as subclasses of the class **Layer**. The **Analog_layer** class specifies the representation of activations, but it still leaves open the procedures that use and update the activation array. **BP_analog_layer** is a subclass of **Analog_layer** that specify these procedures for the back-propagation algorithm. Subclasses of **Analog_layer** may also add new data structures to hold extra internal state such as the error vector in the case of **BP_analog_layer**. The **BP_Analog_layer** class has subclasses for various transfer functions such as **BP_sigmoid_layer** and **BP_linear_layer**.

**Layer** classes also have behavior functions that are called in the course of running the network. For example, one of these functions (**pre_forw_propagate**) initializes the **Layer** for a forward pass, perhaps by clearing its activation vector. After all of the connections coming into it are run, another **Layer** behavior function (**post_forw_propagate**) is called that computes the activation vector from the partial results left by these connections. For example, this function may apply a transfer function such as the sigmoid to the accumulated sum of all the input activations.

These behavior functions can be changed by making a subclass. **BP_analog_layer** leaves open the activation transfer function (or squashing function) and its derivative. Subclasses define new transfer functions to be applied to the activations. A new class of back-propagation layer with a customized transfer function (instead of the default sigmoid) can be created with the following C++ code:

```
class My_new_BP_layer_class : public BP_analog_layer {

  My_new_BP_layer_class(int number_of_units)
    : BP_analog_layer(number_of_units);    // constructor

  void transfer(Fvec *activation) {
    /* apply forward transfer function to my activation vector */
  }

  void d_transfer(Fvec *activation, Fvec *err)  {
    /* apply backward error transfer to err (given activation) */
  }
};
```

A **Connect** class includes two behavior functions: one that transforms activations from the incoming **Layer** into partial results in the outgoing **Layer** (**forw_propagate**) and one that takes outgoing errors and generates partial results in the incoming **Layer** (**back_propagate**).

The structure of a partial result is part of the **Layer** class. The subclasses of **Connect** include: **Bus_connect** (one to one), **Full_connect** (all to all) and **Sparse_connect** (some to some).

Each subclass of **Connect** may contain a set of internal parameters such as the weight matrix in a **BP_full_connect**. Subclasses of **Connect** also specify which pairs of **Layer** subclasses can be connected. When a pair of **Layer** objects are connected, type checking by the C++ compiler insures that the input and output **Layer** subclasses are supported by the **Connect** object.

In order to do its job efficiently, a **Connect** must know something about the internal representation of the layers that are connected. By using C++ overloading, the **Connect** function selected depends not only on the class of **Connect**, but also on the classes of the two layers that are connected. Not all **Connect** classes are defined for all pairs of **Layer** classes. However, **Connects** that convert between **Layer** classes can be utilized to compensate for missing functions.

CLONES allows the user to view layers and connections much like tinker-toy wheels and rods. ANNs are built up by creating **Layer** objects and passing them to the create functions of the desired **Connect** classes. Changing the interconnection pattern does not require any changes to the **Layer** classes or objects and vice-versa.

At the highest level, a **Net** object delineates a subset of a network and controls its training. Operations can be performed on these subsets by calling functions on their **Net** objects. The **Layers** of a **Net** are specified by calling one of **new_input_layer, new_hidden_layer**, or **new_output_layer** on the **Net** object for each **Layer**. Given the **Layers**, the **Connects** that belong to the **Net** are deduced by the **Net_order** objects (see below). **Layer** and **Connect** objects can belong to any number of **Nets**.

The **Net** labels all of its **Layers** as one of input, output or hidden. These labels are used by the **Net_order** objects to determine the order in which the behavior functions of the **Net_parts** are called. For example, a **Net** object contains **Net_order** objects called **forward_pass_order** and **backward_pass_order** that control the execution sequence for a forward or backward pass. The **Net** object also has functions that call a function by the same name on all of its component parts (for example **set_learning_rate**).

When a **Net_order** object is built it scans the connectivity of the **Net**. The rules that relate topology to order of execution are centralized and encapsulated in subclasses of **Net_order**. Changes to the structure of the **Net** are localized to just the code that creates the **Layers** and **Connects**; one does not need to update separate code that contains explict knowledge about the order of evaluation for running a forward or backward pass.

The training procedure is divided into a series of steps, each of which is a call to a function in the **Net** object. At the top level, calling **run_training** on a **Net** performs a complete training run. In addition to calling **pre_training, post_training** behavior functions, it calls **run_epoch** in a loop until the the **next_learning_rate** function returns zero. The **run_epoch** function calls **run_forward** and **run_backward**.

At a lower level there are functions that interface the database(s) of the **Net** object to the **Layers** of the **Net**. For example, **set_input** sets the activations of the input **Layers** for a given pattern number of the database. Another of these sets the error vector of the output layer (**set_error**). Some of these functions, such as **is_correct** evaluate the performance of the **Net** on the current pattern.

In addition to database related functions, the **Net** object also contains useful global variables for all of its components. A pointer to the **Net** object is always passed to all behavior functions of its **Layers** and **Connects** when they are called. One of these variables is a **Param** object that contains a table of parameter names, each with a list of values. These parameters usually come from the command line and/or parameter files. Other variables include: the current pattern, the correct target output, the epoch number, etc.

## 5   Conclusions

CLONES is a useful tool for training ANNs especially when working with large training databases and networks. It runs efficiently on a variety of parallel hardware as well as on UNIX workstations.

**Acknowledgements**

Special thanks to Steve Renals for daring to be the first CLONES user and making significant contributions to the design and implementation. Others who provided valuable input to this work were: Krste Asanović, Steve Omohundro, Jerry Feldman, Heinz Schmidt and Chuck Wooters. Support from the International Computer Science Institute is gratefully acknowledged.

**References**

Asanović, K., Beck, J., Kingsbury, B., Kohn, P., Morgan, N., & Wawrzynek, J. (1991). SPERT: A VLIW/SIMD Microprocessor for Artificial Neural Network Computations. Tech. rep. TR-91-072, International Computer Science Institute.

Beck, J. (1990). The Ring Array Processor (RAP): Hardware. Tech. rep. TR-90-048, International Computer Science Institute.

Bilmes, J. & Kohn, P. (1990). The Ring Array Processor (RAP): Software Architecture. Tech. rep. TR-90-050, International Computer Science Institute.

Bourlard, H. & Morgan, N. (1991). Connectionist approaches to the use of Markov models for continuous speech recognition. In Touretzky, D. S. (Ed.), *Advances in Neural Information Processing Systems*, Vol. 3. Morgan Kaufmann, San Mateo CA.

Kohn, P. & Bilmes, J. (1990). The Ring Array Processor (RAP): Software Users Manual Version 1.0. Tech. rep. TR-90-049, International Computer Science Institute.

Kohn, P. (1991). CLONES: Connectionist Layered Object-oriented NEtwork Simulator. Tech. rep. TR-91-073, International Computer Science Institute.

Morgan, N., Beck, J., Kohn, P., Bilmes, J., Allman, E., & Beer, J. (1990). The RAP: a ring array processor for layered network calculations. In *Proceedings IEEE International Conference on Application Specific Array Processors*, pp. 296–308 Princeton NJ.

Morgan, N., Beck, J., Kohn, P., & Bilmes, J. (1992). Neurocomputing on the RAP. In Przytula, K. W. & Prasanna, V. K. (Eds.), *Digital Parallel Implementations of Neural Networks*. Prentice-Hall, Englewood Cliffs NJ.

## Footnotes

[1]UNIX is a trademark of AT&T

[2]The hardware does not automatically keep the processors in lock step; for example, they may become out of sync because of branches conditioned on the processor's node number or on the data. However, when the processors must communicate with each other through the ring, hardware synchronization automatically occurs. A node that attempts to read before data is ready, or to write when there is already data waiting, will stop executing until the data can be moved.
